# Markov Chain Monte Carlo with People

**Adam N. Sanborn**
Psychological and Brain Sciences
Indiana University
Bloomington, IN 47045
asanborn@indiana.edu

**Thomas L. Griffiths**
Department of Psychology
University of California
Berkeley, CA 94720
tom_griffiths@berkeley.edu

## Abstract

Many formal models of cognition implicitly use subjective probability distributions to capture the assumptions of human learners. Most applications of these models determine these distributions indirectly. We propose a method for directly determining the assumptions of human learners by sampling from subjective probability distributions. Using a correspondence between a model of human choice and Markov chain Monte Carlo (MCMC), we describe a method for sampling from the distributions over objects that people associate with different categories. In our task, subjects choose whether to accept or reject a proposed change to an object. The task is constructed so that these decisions follow an MCMC acceptance rule, defining a Markov chain for which the stationary distribution is the category distribution. We test this procedure for both artificial categories acquired in the laboratory, and natural categories acquired from experience.

## 1 Introduction

Determining the assumptions that guide human learning and inference is one of the central goals of cognitive science. Subjective probability distributions are used to model the degrees of belief that learners assign to hypotheses in many domains, including categorization, decision making, and memory [1, 2, 3, 4]. If the knowledge of learners can be modeled in this way, then exploring this knowledge becomes a matter of asking questions about the nature of their associated probability distributions. A common way to learn about a probability distribution is to draw samples from it. In the machine learning and statistics literature, drawing samples from probability distributions is a major area of research, and is often done using Markov chain Monte Carlo (MCMC) algorithms. In this paper, we describe a method for directly obtaining information about subjective probability distributions, by having people act as elements of an MCMC algorithm.

Our approach is to design a task that will allow us to sample from a particular subjective probability distribution. Much research has been devoted to relating the magnitude of psychological responses to choice probabilities, resulting in mathematical models of these tasks. We point out an equivalence between a model of human choice behavior and an MCMC acceptance function, and use this equivalence to develop a method for obtaining samples from a subjective distribution. In this way we can use the power of MCMC algorithms to explore the knowledge of human learners.

The plan of the paper is as follows. In Section 2, we describe MCMC in general and the Metropolis method and Barker acceptance function in particular. Section 3 describes the experimental task we use to connect human judgments to MCMC. In Section 4, we present an experiment showing that this method can be used to recover trained category distributions from human judgments. Section 5 gives a demonstration of our MCMC method applied to recovering natural categories of animal shape. Section 6 summarizes the results and discusses some implications.

## 2  Markov chain Monte Carlo

Models of physical phenomena used by scientists are often expressed in terms of complex probability distributions over different events. Generating samples from these distributions can be an efficient way to determine their properties, indicating which events are assigned high probabilities and providing a way to approximate various statistics of interest. Often, the distributions used in these models are difficult to sample from, being defined over large state spaces or having unknown normalization constants. Consequently, a great deal of research has been devoted to developing sophisticated Monte Carlo algorithms that can be used to generate samples from complex probability distributions. One of the most successful methods of this kind is Markov chain Monte Carlo. An MCMC algorithm constructs a Markov chain that has the target distribution, from which we want to sample, as its stationary distribution. This Markov chain can be initialized with any state, being guaranteed to converge to its stationary distribution after many iterations of stochastic transitions between states. After convergence, the states visited by the Markov chain can be used similarly to samples from the target distribution (see [5] for details).

The canonical MCMC algorithm is the Metropolis method [6], in which transitions between states have two parts: a proposal distribution and an acceptance function. Based on the current state, a candidate for the next state is sampled from the proposal distribution. The acceptance function gives the probability of accepting this proposal. If the proposal is rejected, then the current state is taken as the next state. A variety of acceptance functions guarantee that the stationary distribution of the resulting Markov chain is the target distribution [7]. If we assume that the proposal distribution is symmetric, with the probability of proposing a new state $x^*$ from the current state $x$ being the same as the probability of proposing $x$ from $x^*$, we can use the Barker acceptance function [8], giving

$$A(x^*; x) \quad = \quad \frac{\pi(x^*)}{\pi(x^*) + \pi(x)} \tag{1}$$

for the acceptance probability, where $\pi(x)$ is the probability of $x$ under the target distribution.

## 3  An acceptance function from human behavior

While our approach can be applied to any subjective probability distribution, our experiments focused on sampling from the distributions over objects associated with different categories. Categories are central to cognition, reflecting our knowledge of the structure of the world, supporting inferences, and serving as the basic units of thought. The way people group objects into categories has been studied extensively, producing a number of formal models of human categorization [3, 4, 9, 10, 11], almost all of which can be interpreted as defining a category as a probability distribution over objects [4]. In this section, we consider how to lead people to choose between two objects in a way that would correspond to a valid acceptance function for an MCMC algorithm with the distribution over objects associated with a category as its target distribution.

### 3.1  A Bayesian analysis of a choice task

Consider the following task. You are shown two objects, $x_1$ and $x_2$, and told that one of those objects comes from a particular category, $c$. You have to choose which object you think comes from that category. How should you make this decision?

We can analyze this choice task from the perspective of a rational Bayesian learner. The choice between the objects is a choice between two hypotheses: The first hypothesis, $h_1$, is that $x_1$ is drawn from the category distribution $p(x|c)$ and $x_2$ is drawn from $g(x)$, an alternative distribution that governs the probability of other objects appearing on the screen. The second hypothesis, $h_2$, is that $x_1$ is from the alternative distribution and $x_2$ is from the category distribution. The posterior probability of the first hypothesis given the data is determined via Bayes' rule,

$$
\begin{aligned}
p(h_1|x_1, x_2) \quad &= \quad \frac{p(x_1, x_2|h_1)p(h_1)}{p(x_1, x_2|h_1)p(h_1) + p(x_1, x_2|h_2)p(h_2)} \\
&= \quad \frac{p(x_1|c)g(x_2)p(h_1)}{p(x_1|c)g(x_2)p(h_1) + p(x_2|c)g(x_1)p(h_2)}
\end{aligned}
\tag{2}
$$

where we use the category distribution $p(x|c)$ and its alternative $g(x)$ to calculate $p(x_1, x_2|h)$.

We will now make two assumptions. The first assumption is that the prior probabilities of the hypotheses are the same. Since there is no a priori reason to favor one of the objects over the other, this assumption seems reasonable. The second assumption is that the probabilities of the two stimuli under the alternative distribution are approximately equal, with $g(x_1) \approx g(x_2)$. If people assume that the alternative distribution is uniform, then the probabilities of the two stimuli will be exactly equal. However, the probabilities will still be roughly equal under the weaker assumption that the alternative distribution is fairly smooth and $x_1$ and $x_2$ differ by only a small amount relative to the support of that distribution. With these assumptions Equation 2 becomes

$$p(h_1|x_1, x_2) \approx \frac{p(x_1|c)}{p(x_1|c) + p(x_2|c)} \tag{3}$$

with the posterior probability of $h_1$ being set by the probabilities of $x_1$ and $x_2$ in that category.

## 3.2 From a task to an acceptance function

The Bayesian analysis of the task described above results in a posterior probability of $h_1$ (Equation 3) which has a similar form to the Barker acceptance function (Equation 1). If we return to the context of MCMC, and assume that $x_1$ is the proposal $x^*$ and $x_2$ the current state $x$, and that people choose $x_1$ with probability equal to the posterior probability of $h_1$, then $x^*$ is chosen with probability

$$A(x^*; x) = \frac{p(x^*|c)}{p(x^*|c) + p(x|c)} \tag{4}$$

being the Barker acceptance function for the target distribution $\pi(x) = p(x|c)$. This equation has a long history as a model of human choice probabilities, where it is known as the Luce choice rule or the ratio rule [12, 13]. This rule has been shown to provide a good fit to human data when people choose between two stimuli based on a particular property [14, 15, 16]. It corresponds to a situation in which people choose alternatives based on their relative probabilities, a common behavior known as probability matching [17]. The Luce choice rule has also been used to convert psychological response magnitudes into response probabilities in many models of cognition [11, 18, 19, 20, 21].

## 3.3 A more flexible response rule

Probability matching can be a good description of the data, but subjects have been shown to produce behavior that is more deterministic [17]. Several models of categorization have been extended in order to account for this behavior [22] by using an exponentiated version of Equation 4 to map category probabilities onto response probabilities,

$$A(x^*; x) = \frac{p(x^*|c)^\gamma}{p(x^*|c)^\gamma + p(x|c)^\gamma} \tag{5}$$

where $\gamma$ raises each term on the right side of Equation 4 to a constant. This response rule can be derived by applying a soft threshold to the log odds of the two hypotheses (a sigmoid function with a gain of $\gamma$). As $\gamma$ increases the hypothesis with higher posterior probability will be chosen more often. By equivalence to the Barker acceptance function, this response rule defines a Markov chain with stationary distribution

$$\pi(x) \propto p(x|c)^\gamma. \tag{6}$$

Thus, using the weaker assumptions of Equation 5 as a model of human behavior, we can estimate the category distribution $p(x|c)$ up to a constant exponent. This estimate will have the same modes and ordering of variances on the variables, but the actual values of the variances will differ.

## 3.4 Summary

Based on the results in this section, we can define a simple method for drawing samples from a category distribution using MCMC. On each trial, a proposal is drawn from a symmetric distribution. A person chooses between the current state and the proposal to select the new state. Assuming that people's choice behavior follows the Luce choice rule, the stationary distribution of the Markov chain is the category distribution. The states of the chain are samples from the category distribution, which provide information about the mental representation of that category.

# 4 Testing the MCMC algorithm with known categories

To test whether the procedure outlined in the previous section will produce samples that accurately reflect people's mental representations, we trained people on a variety of category distributions and attempted to recover those distributions using MCMC. A simple one-dimensional categorization task was used, with the height of schematic fish (see Figure 1) being the dimension along which category distributions were defined. Subjects were trained on two categories of fish height – a uniform distribution and a Gaussian distribution – being told that they were learning to judge whether a fish came from the ocean (the uniform distribution) or a fish farm (the Gaussian distribution). Four between-subject conditions tested different means and variances for the Gaussian distributions. Once subjects were trained, we collected MCMC samples for the Gaussian distributions by asking subjects to judge which of two fish came from the fish farm.

## 4.1 Method

Fifty subjects were recruited from the university community via a newspaper advertisement. Data from one subject was discarded for not finishing the experiment, data from another was discarded because the chains reached a boundary, and the data of eight others were discarded because their chains did not cross (more detail below). There were ten observers in each between-subject condition. Each subject was paid \$4 for a 35 minute session. The experiment was presented on a Apple iMac G5 controlled by a script running in Matlab using PsychToolbox extensions [23, 24]. Observers were seated approximately 44 cm away from the display.

Each subject was trained to discriminate between two categories of fish: ocean fish and fish farm fish. Subjects were instructed, "Fish from the ocean have to fend for themselves and as a result they have an equal probability of being any size. In contrast, fish from the fish farm are all fed the same amount of food, so their sizes are similar and only determined by genetics." These instructions were meant to suggest that the ocean fish were drawn from a uniform distribution and the fish farm fish were drawn from a Gaussian distribution. The mean and the standard deviation of the Gaussian were varied in four between-subject conditions, resulting from crossing two levels of the mean, $\mu = 3.66$ cm and $\mu = 4.72$ cm, with two levels of the standard deviation, $\sigma = 3.1$ mm and $\sigma = 1.3$ mm. The uniform distribution was the same across training distributions and was bounded at 2.63 cm and 5.76 cm.

The stimuli were a modified version of the fish used in [25]. The fish were constructed from three ovals, two gray and one black, and a circle on a black background. Fish were all 9.1 cm long with heights drawn from the Gaussian and uniform distributions in training. Examples of the smallest and largest fish are shown in Figure 1. During the the MCMC trials, the range of possible fish heights was expanded to be from 0.3 mm to 8.35 cm.

Subjects saw two types of trials. In a training trial, either the uniform or Gaussian distribution was selected with equal probability, and a single sample was drawn from the selected distribution. The sampled fish was shown to the subject, who chose which distribution produced the fish. Feedback was then provided on the accuracy of this choice. In an MCMC trial, two fish were presented on the screen. Subjects chose which of the two fish came from the Gaussian distribution. Neither fish had been sampled from the Gaussian distribution. Instead, one fish was the state of a Markov chain and the other fish was the proposal. The state and proposal were unlabeled and they were randomly assigned to either the left or right side of the screen. Three MCMC chains were interleaved during the MCMC trials. The start states of the chains were chosen to be 2.63 cm, 4.20 cm, and 5.76 cm. Relative to the training distributions, the start states were overdispersed, facilitating assessment of

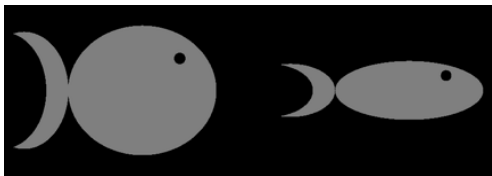

Figure 1: Examples of the largest and smallest fish stimuli presented to subjects during training. The relative size of the fish stimuli are shown here; true display sizes are given in the text.

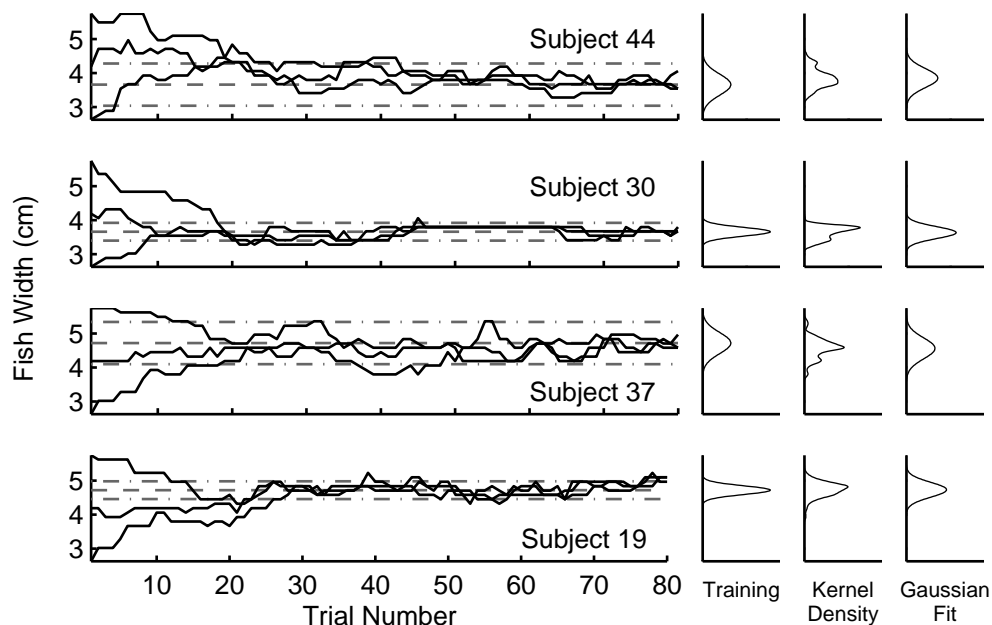

Figure 2: The four rows are subjects from each of the between-subject conditions. The panels in the first column show the behavior of the three Markov chains per subject. The black lines represent the states of the Markov chains, the dashed line is the mean of the Gaussian training distribution, and the dot-dashed lines are two standard deviations from the mean. The second column shows the densities of the training distributions. These training densities can be compared to the MCMC samples, which are described by their kernel density estimates and Gaussian fits in the last two columns.

convergence. The proposal was chosen from a symmetric discretized pseudo-Gaussian distribution with a mean equal to the current state. The probability of proposing the current state was set to zero.

The experiment was broken up into blocks of training and MCMC trials, beginning with 120 training trials, followed by alternating blocks of 60 MCMC trials and 60 training trials. Training and MCMC trials were interleaved to keep subjects from forgetting the training distributions. A block of 60 test trials, identical to the training trials but without feedback, ended the experiment.

## 4.2   Results

Subjects were excluded if their chains did not converge to the stationary distribution or if the state of any chain reached the edge of the parameter range. We used a heuristic for determining convergence: every chain had to cross another chain.[1] Figure 2 shows the chains from four subjects, one from each of the between-subject conditions. Most subjects took approximately 20 trials to produce the first crossing in their chains, so these trials were discarded and the remaining 60 trials from each chain were pooled and used in further analyses.

The distributions on the right hand side of Figure 2 show the training distribution, best fit Gaussian to the MCMC samples, and kernel density estimate based on the MCMC samples. The distributions estimated for the subjects shown in this figure match well with the training distribution. The mean, $\mu$, and standard deviation, $\sigma$, were computed from the MCMC samples produced by each subject. The average of these estimates for each condition is shown in Figure 3. As predicted, $\mu$ was higher for subjects trained on Gaussians with higher means, and $\sigma$ was higher for subjects trained on Gaussians with higher standard deviations. These differences were statistically significant, with a one-tailed Student's $t$-test for independent samples giving $t(38) = 7.36, p < 0.001$ and $t(38) = 2.01, p < 0.05$ for $\mu$ and $\sigma$ respectively. The figure also shows that the means of the MCMC samples corresponded well with the actual means of the training distributions. The standard deviations of the samples tended to be higher than the training distributions, which could be a consequence of either perceptual

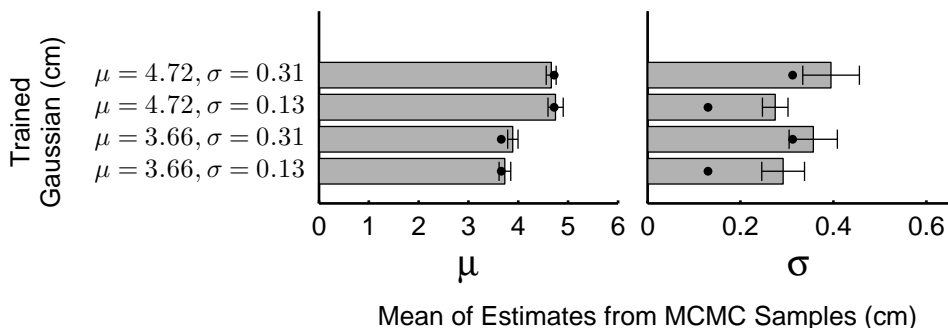

Figure 3: The bar plots show the mean of $\mu$ and $\sigma$ across the MCMC samples produced by subjects in all four training conditions. Error bars are one standard error. The black dot indicates the actual value of $\mu$ and $\sigma$ for each condition.

noise (increasing the effective variation in stimuli associated with a category) or choices being made in a way consistent with the exponentiated choice rule with $\gamma < 1$.

## 5  Investigating the structure of natural categories

The previous experiment provided evidence that the assumptions underlying the MCMC method are approximately correct, as the samples recovered by the method matched the training distribution. Now we will demonstrate this method in a much more interesting case: sampling from subjective probability distributions that have been built up from real-world experience. The natural categories of the shapes of giraffes, horses, cats, and dogs were explored in a nine-dimensional stick figure space [26]. The responses of a single subject are shown in Figure 4. For each animal, three Markov chains were started from different states. The three starting states were the same between animal conditions. Figure 4B shows the chains converging for the giraffe condition. The different animal conditions converged to different areas of the parameter space (Figure 4C) and the means across samples produced stick figures that correspond well to the tested categories (Figure 4D).

## 6  Summary and conclusion

We have developed a Markov chain Monte Carlo method for sampling from a subjective probability distribution. This method allows a person to act as an element of an MCMC algorithm by constructing a task for which choice probabilities follow a valid acceptance function. By choosing between the current state and a proposal, people produce a Markov chain with a stationary distribution matching their mental representation of a category. The results of our experiment indicate that this method accurately uncovers differences in mental representations that result from training people on categories with different structures. In addition, we explored the subjective probability distributions of natural animal shapes in a multidimensional parameter space.

This method is a complement to established methods such as classification images [27]. Our method estimates the subjective probability distribution, while classification images estimate the decision boundary between two classes. Both methods can contribute to the complete picture of how people make categorization decisions. The MCMC method corresponds most closely to procedures for gathering typicality ratings in categorization research. Typicality ratings are used to determine which objects are better examples of a category than other objects. Our MCMC method yields the same information, but provides a way to efficiently do so when the category distribution is concentrated in a small region of a large parameter space. Testing a random subset of objects from this type of space will result in many uninformative trials. MCMC's use of previous responses to select new test trials is theoretically more efficient, but future work is needed to empirically validate this claim.

Our MCMC method provides a way to explore the subjective probability distributions that people associate with categories. Similar tasks could be used to investigate subjective probability distributions in other settings, providing a valuable tool for testing probabilistic models of cognition. The general principle of identifying connections between models of human performance and machine learning algorithms can teach us a great deal about cognition. For instance, Gibbs sampling could

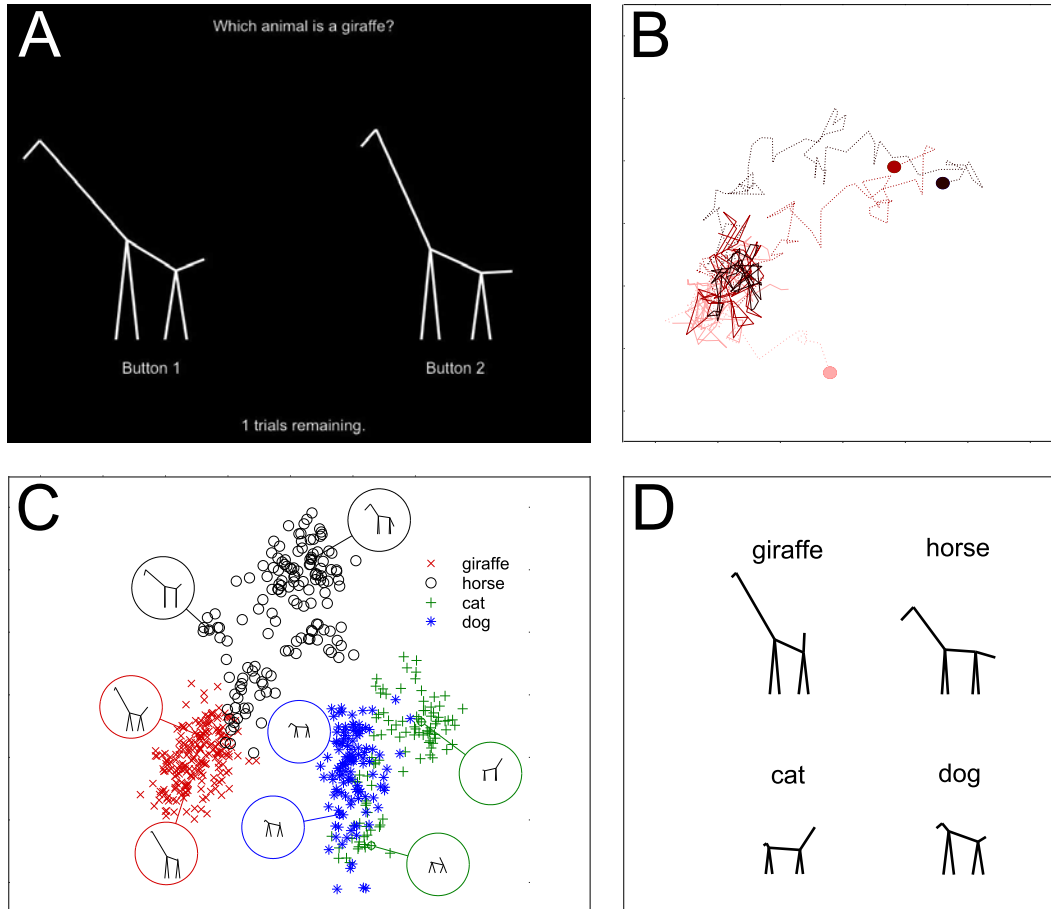

Figure 4: Task and results for an experiment exploring natural categories of animals using stick figure stimuli. (A) Screen capture from the experiment, where people make a choice between the current state of the Markov chain and a proposed state. (B) States of the Markov chain for the subject when estimating the distribution for giraffes. The nine-dimensional space characterizing the stick figures is projected onto the two dimensions that best discriminate the different animal distributions using linear discriminant analysis. Each chain is a different color and the start states of the chains are indicated by the filled circle. The dotted lines are samples that were discarded to ensure that the Markov chains had converged, and the solid lines are the samples that were retained. (C) Samples from distributions associated with all four animals for the subject, projected onto the same plane used in B. Two samples from each distribution are displayed in the bubbles. The samples capture the similarities and differences between the four categories of animals, and reveal the variation in the members of those categories.(D) Mean of the samples for each animal condition.

be used to generate samples from a distribution, if a clever method for inducing people to sample from conditional distributions could be found. Using people as the elements of a machine learning algorithm is a virtually unexplored area that should be exploited in order to more efficiently test hypotheses about the knowledge that guides human learning and inference.

## Footnotes

[1]Many heuristics have been proposed for assessing convergence. The heuristic we used is simple to apply in a one-dimensional state space. It is a necessary, but not sufficient, condition for convergence.

## References

[1] M. Oaksford and N. Chater, editors. *Rational models of cognition*. Oxford University Press, 1998.

[2] N. Chater, J. B. Tenenbaum, and A. Yuille. Special issue on "Probabilistic models of cognition". *Trends in Cognitive Sciences*, 10(7), 2006.

[3] J. R. Anderson. *The adaptive character of thought*. Erlbaum, Hillsdale, NJ, 1990.

[4] F. G. Ashby and L. A. Alfonso-Reese. Categorization as probability density estimation. *Journal of Mathematical Psychology*, 39:216–233, 1995.

[5] W.R. Gilks, S. Richardson, and D. J. Spiegelhalter, editors. *Markov Chain Monte Carlo in Practice*. Chapman and Hall, Suffolk, 1996.

[6] A. W. Metropolis, A. W. Rosenbluth, M. N. Rosenbluth, A. H. Teller, and E. Teller. Equations of state calculations by fast computing machines. *Journal of Chemical Physics*, 21:1087–1092, 1953.

[7] W. K. Hastings. Monte Carlo sampling methods using Markov chains and their applications. *Biometrika*, 57:97–109, 1970.

[8] A. A. Barker. Monte Carlo calculations of the radial distribution functions for a proton-electron plasma. *Australian Journal of Physics*, 18:119–133, 1965.

[9] S. K. Reed. Pattern recognition and categorization. *Cognitive Psychology*, 3:393–407, 1972.

[10] D. L. Medin and M. M. Schaffer. Context theory of classification learning. *Psychological Review*, 85:207–238, 1978.

[11] R. M. Nosofsky. Attention, similarity, and the identification-categorization relationship. *Journal of Experimental Psychology: General*, 115:39–57, 1986.

[12] R. D. Luce. Detection and recognition. In R. D. Luce, R. R. Bush, and E. Galanter, editors, *Handbook of Mathematical Psychology, Volume 1*, pages 103–190. John Wiley and Sons, Inc., New York and London, 1963.

[13] R. N. Shepard. Stimulus and response generalization: A stochastic model relating generalization to distance in psychological space. *Psychometrika*, 22:325–345, 1957.

[14] R. A. Bradley. Incomplete block rank analysis: On the appropriateness of the model of a method of paired comparisons. *Biometrics*, 10:375–390, 1954.

[15] F. R. Clarke. Constant-ratio rule for confusion matrices in speech communication. *The Journal of the Acoustical Society of America*, 29:715–720, 1957.

[16] J. W. Hopkins. Incomplete block rank analysis: Some taste test results. *Biometrics*, 10:391–399, 1954.

[17] N. Vulkan. An economist's perspective on probability matching. *Journal of Economic Surveys*, 14:101–118, 2000.

[18] F. G. Ashby. *Multidimensional models of perception and cognition*. Erlbaum, Hillsdale, NJ, 1992.

[19] R. M. Nosofsky. Attention and learning processes in the identification and categorization of integral stimuli. *Journal of Experimental Psychology: Learning, Memory, and Cognition*, 13:87–108, 1987.

[20] G. C. Oden and D. W. Massaro. Integration of featural information in speech perception. *Psychological Review*, 85:172–191, 1978.

[21] J. L. McClelland and J. L. Elman. The TRACE model of speech perception. *Cognitive Psychology*, 18:1–86, 1986.

[22] F. G. Ashby and W. T. Maddox. Relations between prototype, exemplar, and decision bound models of categorization. *Journal of Mathematical Psychology*, 37:372–400, 1993.

[23] D. H. Brainard. The psychophysics toolbox. *Spatial Vision*, 10:433–436, 1997.

[24] D. G. Pelli. The VideoToolbox software for visual psychophysics: Transforming numbers into movies. *Spatial Vision*, 10:437–442, 1997.

[25] J. Huttenlocher, L. V. Hedges, and J. L. Vevea. Why do categories affect stimulus judgment? *Journal of Experimental Psychology: General*, 129:220–241, 2000.

[26] C. Olman and D. Kersten. Classification objects, ideal observers, and generative models. *Cognitive Science*, 28:227–239, 2004.

[27] A. J. Ahumada and J. Lovell. Stimulus features in signal detection. *Journal of the Acoustical Society of America*, 49:1751–1756, 1971.

